# Sigma-Pi Learning: On Radial Basis Functions and Cortical Associative Learning

**Bartlett W. Mel**       **Christof Koch**
Computation and Neural Systems Program
Caltech, 216-76
Pasadena, CA 91125

## ABSTRACT

The goal in this work has been to identify the neuronal elements of the cortical column that are most likely to support the learning of nonlinear associative maps. We show that a particular style of network learning algorithm based on locally-tuned receptive fields maps naturally onto cortical hardware, and gives coherence to a variety of features of cortical anatomy, physiology, and biophysics whose relations to learning remain poorly understood.

## 1   INTRODUCTION

Synaptic modification is widely believed to be the brain's primary mechanism for long-term information storage. The enormous practical and theoretical importance of biological synaptic plasticity has stimulated interest among both experimental neuroscientists and neural network modelers, and has provided strong incentive for the development of computational models that can both explain and predict.

We present here a model for the synaptic basis of associative learning in cerebral cortex. The main hypothesis of this work is that the principal output neurons of a cortical association area learn functions of their inputs as locally-generalizing lookup tables. As abstractions, locally-generalizing learning methods have a long history in statistics and approximation theory (see Atkeson, 1989; Barron & Barron,

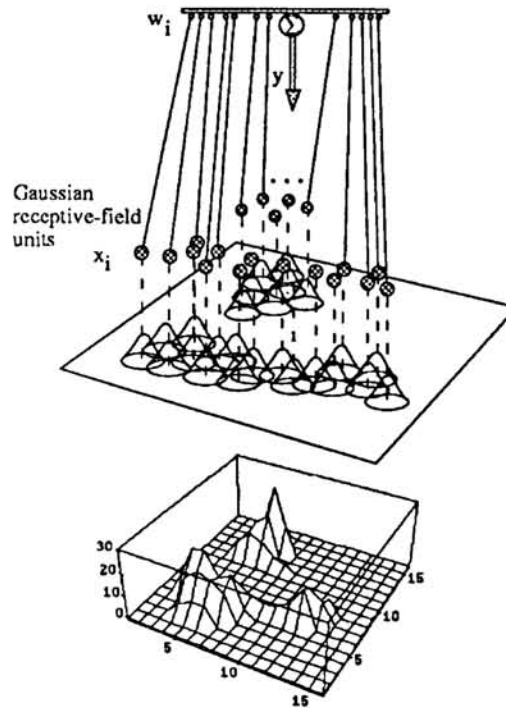

**Figure 1:** A Neural Lookup Table. A nonlinear function of several variables may be decomposed as a weighted sum over a set of localized "receptive fields" units.

1988). Radial Basis Function (RBF) methods are essentially similar (see Broomhead & Lowe, 1988) and have recently been discussed by Poggio and Girosi (1989) in relation to regularization theory. As is standard for network learning problems, locally-generalizing methods involve the learning of a map $f(\underline{x}) : \underline{x} \mapsto y$ from example $(\underline{x}, y)$ pairs. Rather than operate directly on the input space, however, input vectors are first "decoded" by a population of "receptive field" units with centers $\xi_i$ that each represents a local, often radially-symmetric, region in the input space. Thus, an output unit computes its activation level $y = \sum_i w_i g(x - \xi_i)$, where $g$ defines a "radial basis function", commonly a Gaussian, and $w_i$ is its weight (Fig. 1). The learning problem can then be characterized as one of finding weights $\underline{w}$ that minimize the mean squared error over the $N$ element training set. Learning schemes of this type lend themselves directly to very simple Hebb-type rules for synaptic modification since the initially nonlinear learning problem is transformed into a linear one in the unknown parameters $\underline{w}$ (see Broomhead & Lowe, 1988).

Locally-generalizing learning algorithms as neurobiological models date at least to Albus (1971) and Marr (1969, 1970); they have also been explored more recently by a number of workers with a more pure computational bent (Broomhead & Lowe, 1988; Lapedes & Farber, 1988; Mel, 1988, 1989; Miller, 1988; Moody, 1989; Poggio & Girosi, 1989).

## 2    SIGMA-PI LEARNING

Unlike the classic thresholded linear unit that is the mainstay of many current connectionist models, the output of a *sigma-pi* unit is computed as a sum of contributions from a set of independent multiplicative clusters of input weights (adapted from Rumelhart & McClelland, 1986): $y = \sigma(\sum_j w_j c_j)$, where $c_j = \prod_i v_i x_i$ is the product of weighted inputs to cluster $j$, $w_j$ is the weight on cluster $j$ as a whole, and $\sigma$ is an optional thresholding nonlinearity applied to the sum of total cluster activity. During learning, the output may also by clamped by an unconditioned teacher input, i.e. such that $y = t_i(\underline{x})$. Units of this general type were first proposed by Feldman & Ballard (1982), and have been used occasionally by other connectionist modelers, most commonly to allow certain inputs to gate others or to allow the activation of one unit to control the strength of interconnection between two other units (Rumelhart & McClelland, 1986). The use of *sigma-pi* units as function lookup tables was suggested by Feldman & Ballard (1982), who cited a possible relevance to local dendritic interactions among synaptic inputs (see also Durbin & Rumelhart, 1989).

In the present work, the specific nonlinear interaction among inputs to a *sigma-pi* cluster is not of primary theoretical importance. The crucial property of a cluster is that its output should be AND-like, i.e. selective for the simultaneous activity of *all* of its $k$ input lines[1].

### 2.1    NETWORK ARCHITECTURE

We assume an underlying $d$-dimensional input space $\mathcal{X} \in R^d$ over which functions are to be learned. Vectors in $\mathcal{X}$ are represented by a population X of $N$ units whose state is denoted by $\underline{x} \in R^N$. Within X, each of the $d$ dimensions of $\mathcal{X}$ is individually value-coded, i.e. consists of a set of units with gaussian receptive fields distributed in overlapping fashion along the range of allowable parameter values, for example, the angle of a joint, or the orientation of a visual stimulus at a specific retinal location. (A more biologically realistic case would allow for individual units in X to have *multi*-dimensional gaussian receptive fields, for example a 4-d visual receptive field encoding retinal x and y, edge orientation, and binocular disparity.)

We assume a map $t(\underline{x}) : \underline{x} \mapsto \underline{y}$ is to be learned, where the components of $\underline{y} \in R^M$ are represented by an output population Y of $M$ units. According to the familiar single-layer feedforward network learning paradigm, X projects to Y via an "associational" pathway with modifiable synapses. We consider the task of a single output unit $y_i$ (hereafter denoted by $y$), whose job is to estimate the underlying teacher function $t_i(\underline{x}) : \underline{x} \mapsto y$ from examples. Output unit $y$ is assumed to have access to the entire input vector $\underline{x}$, and a single unconditioned teacher input $t_i$. We further assume that

all possible clusters $c_j$ of size 1 through $k = k_{max}$ pre-exist in $y$'s dendritic field, with cluster weights $w_j$ initially set to 0, and input weights $v_i$ within each cluster set equal to 1. Following from our assumption that each of the input lines $x_i$ represents a 1-dimensional gaussian receptive field in $\mathcal{X}$, a multiplicative cluster of $k$ such inputs can yield a $k$-dimensional receptive field in $\mathcal{X}$ that may then be weighted. In this way, a *sigma-pi* unit can directly implement an RBF decomposition over $\mathcal{X}$. Additionally, since a *sigma-pi* unit is essentially a massively parallel lookup table with clusters as stored table entries, it is significant that the *sigma-pi* function is inherently *modular*, such that groups of *sigma-pi* units that receive the same teacher signal can, by simply adding their outputs, act as a single much larger virtual *sigma-pi* unit with correspondingly increased table capacity[2]. A neural architecture that allows system storage capacity to be multiplied by a factor of $k$ by growing $k$ neurons in the place of one, is one that should be strongly preferred by biological evolution.

## 2.2  THE LEARNING RULE

The cluster weights $w_j$ are modified during training according to the following self-normalizing Hebb rule:

$$\dot{w}_j = \alpha \, c_{jp} \, t_p - \beta w_j,$$

where $\alpha$ and $\beta$ are small positive constants, and $c_{jp}$ and $t_p$ are, respectively, the $j$th cluster response and teacher signal in state $p$. The steady state of this learning rule occurs when $w_j = \frac{\alpha}{\beta} <c_j t>$, which tries to maximize the correlation[3] of cluster output and teacher signal over the training set, while minimizing total synaptic weight for all clusters. The inputs weights $v_i$ are unmodified during learning, representing the degree of cluster membership for each input line.

We briefly note that because this Hebb-type learning rule is truly local, i.e. depends only upon activity levels available directly at a synapse to be modified, it may be applied transparently to a group of neurons driven by the same global teacher input (see above discussion of *sigma-pi* modularity). Error-correcting rules that modify synapses based on a difference between desired vs. actual neural output do not share this property.

## 3  TOWARD A BIOLOGICAL MODEL

In the remainder of this paper we examine the hypothesis that *sigma-pi* units underlie associative learning in cerebral cortex. To do so, we identify the six essential elements of the *sigma-pi* learning scheme and discuss the evidence for each: i) a population of output neurons, ii) a focal teacher input, iii), a diffuse association input, iv) Hebb-type synaptic plasticity, v) local dendritic multiplication (or thresholding), and vi) a cluster reservoir.

Following Eccles (1985), we concern ourselves here with the cytoarchitecture of "generic" association cortex, rather than with the more specialized (and more often studied) primary sensory and motor areas. We propose the cortical circuit of fig.

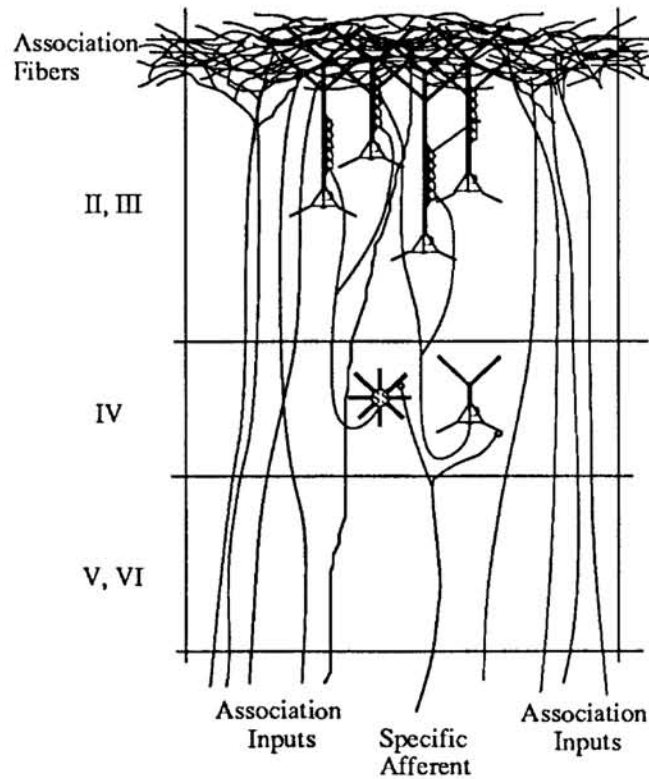

Association
Fibers

II, III

IV

V, VI

Association
Inputs        Specific      Association
Afferent       Inputs

**Figure 2:** Elements of the cortical column in a generic association cortex.

2 to contain all of the basic elements necessary for associative learning, closely paralleling the accounts of Marr (1970) and Eccles (1985) at this level of description. We limit our focus to the cortically-projecting "output" pyramids of layers II and III, which are posited to be *sigma-pi* units. These cells are a likely locus of associative learning as they are well situated to receive both teacher and associational input pathways. With reference to the modularity property of *sigma-pi* learning (sec. 2.1), we interpret the aggregates of layer II/III pyramidal cells whose apical dendrites rise toward the cortical surface in tight clumps (on the order of 100 cells, Peters, 1989), as a single virtual *sigma-pi* unit.

## 3.1   THE TEACHER INPUT

We tentatively define the "teacher" input to an association area to be those inputs that terminate primarily in layer IV onto spiny stellate cells or small pyramidal cells. Lund et al. (1985) points out that spiny stellate cells are most numerous in primary sensory areas, but that the morphologically similar class of small pyramidal cells in layer IV seem to mimic the spiny stellates in their local, vertically oriented excitatory axonal distributions. The layer IV spiny stellates are known to project primarily up (but also down) a narrow vertical cylinder in which they sit, probably making powerful "cartridge" synapses onto overlying pyramidal cells. These excitatory interneurons are presumably capable of strongly deplorarizing entire output cells (Szentagothai, 1977), thus providing the needed unit-wide teacher signals to the output neurons. We therefore assume this teacher pathway plays a role analogous to the presumed role of cerebellar climbing fibers (Albus, 1971; Marr,

1969) The inputs to layer IV can be of both thalamic and/or cortical origin.

## 3.2   THE ASSOCIATIONAL INPUT

A second major form of extrinsic excitatory input with access to layer II/III pyramidal cells is the massive system of horizontal fibers in layer I. The primary source of these fibers is currently believed to be long range excitatory association fibers from both other cortical and subcortical areas (Jones, 1981). In accordance with Marr (1970) and Eccles (1985), we interpret this system of horizontal fibers, which virtually permeates the dendritic fields of the layer II/III pyramidal cells, as the primary *conditioned* input pathway at which cortical associative learning takes place. There is evidence that an individual layer I fibers can make excitatory synapses on apical dendrites of pyramidal cells across an area of cortex 5-6mm in diameter (Szentagothai, 1977).

## 3.3   HEBB RULES, MULTIPLICATION, AND CLUSTERING

The process of cluster formation in *sigma-pi* learning is driven by a local Hebb-type rule. Long term Hebb-type synaptic modification has been demonstrated in several cortical areas, dependent only upon local post-synaptic depolarization (Kelso et al., 1986), and thought to be mediated by the the voltage-dependent NMDA channel (see Brown et al., 1988). In addition to the standard tendency for LTP with pre- and post-synaptic correlation, *sigma-pi* learning implicitly specifies cooperation among pre-synaptic units, in the sense that the largest increase in cluster weight $w_j$ occurs when all inputs $x_i$ to a cluster are simultaneously and strongly active. This type of cooperation among pre-synaptic inputs should follow directly from the assumption that local post-synaptic depolarization is the key ingredient in the induction of LTP. In other words, like-activated synaptic inputs must inevitably contribute to each other's enhancement during learning to the extent they are clustered on a post-synaptic dendrite. This type of cooperativity in learning gives key importance to *dendritic space* in neural learning, and has not until very recently been modelled at a biophysical level (T. Brown, pers. comm; J. Moody, pers. comm.).

In addition to its possible role in enhancing like-activated synaptic clusters however, the NMDA channel may be hypothesized to simultaneously underlie the "multiplicative" interaction among neighboring inputs needed for ensuring cluster-selectivity in *sigma-pi* learning. Thus, if sufficiently endowed with NMDA channels, cortical pyramidal cells could respond highly selectively to associative input "vectors" whose active afferents are spatially clumped, rather than scattered uniformly, across the dendritic arbor. The possibility that dendritic computations could include local multiplicative nonlinearities is widely accepted (e.g. Shepherd et al., 1985; Koch et al., 1983).

## 3.4   A VIRTUAL CLUSTER RESERVOIR

The abstract definition of *sigma-pi* learning specifies that all possible clusters $c_j$ of size $1 < k < k_{max}$ pre-exist on the "dendrites" of each virtual *sigma-pi* unit (which we have previously proposed to consist of a vertically aggregated clump of 100

pyramidal cells that receive the same teacher input from layer 4). During learning, the weight on each cluster is governed by a simple Hebb rule. Since the number of *possible* clusters of size $k$ overwhelms total available dendritic space for even small $k^4$, it must be possible to *create* a cluster when it is needed. We propose that the complex 3-d mesh of axonal and dendritic arborizations in layer 1 are ideal for maximizing the probability that arbitrary (small) subsets of association axons cross *near* to each other in space at some point in their collective arborizations. Thus, we propose that the tangle of axons within a dendrite's receptive field gives rise to an enormous set of *almost*-clusters, poised to "latch" onto a post-synaptic dendrite when called for by a Hebb-type learning rule. This geometry of pre- and post-synaptic interface is to be strongly contrasted with the architecture of cerebellum, where the afferent "parallel" fibers have no possibility of clustering on post-synaptic dendrites.

Known biophysical mechamisms for the sprouting and guidance of growth cones during development, in some cases driven by neural activity seem well suited to the task of cluster formation over small distances in the adult brain.

## 4    CONCLUSIONS

The locally-generalizing, table-based *sigma-pi* learning scheme is a parsimonious mechanisms that can account for the learning of nonlinear associative maps in cerebral cortex. Only a single layer of excitatory synapses is modified, under the control of a Hebb-type learning rule. Numerous open questions remain however, for example the degree to which clusters of active synapses scattered across a pyramidal dendritic tree can function independently, providing the necessary AND-like selectivity.

### Acknowledgements

Thanks are due to Ojvind Bernander, Rodney Douglas, Richard Durbin, Kamil Grajski, David Mackay, and John Moody for numerous helpful discussions. We acknowledge support from the Office of Naval Research, the James S. McDonnell Foundation, and the Del Webb Foundation.

## Footnotes

[1] A local threshold function can act as an AND in place of a multiplication, and for purposes of biological modeling, is a more likely dendritic mechanism than pure multiplication. In continuing work, we are exploring the more detailed interactions between Hebb-type learning rules and various post-synaptic nonlinearities, specifically the NMDA channel, that could underlie a multiplication relation among nearby inputs.

[2] This assumes the global thresholding nonlinearity $\sigma$ is weak, i.e. has an extended linear range.

[3] Strictly speaking, the average product.

[4] For example, assume a 3-d learning problem and clusters of size $k = 3$; with 100 afferents per input dimension, there are $100^3 = 10^6$ possible clusters. If we assume 5,000 available association synapses per pyramidal cell, there is dendritic space for at most 166,000 clusters of size 3.

### References

Albus, J.S. A theory of cerebellar function. *Math. Biosci.*, 1971, *10*, 25-61.

Atkeson, C.G. Using associative content-addressable memories to control robots, MIT A.I. Memo 1124, September 1989.

Barron, A.R. & Barron, R.L. Statistical learning networks: a unifying view. Presented at the *1988 Symposium on the Interface: Statistics and Computing Science*, Reston, Virginia.

Bliss, T.V.P. & Lømo, T. Long-lasting potentiation of synaptic transmission in the dentate area of the anaesthetized rabbit following stimulation of the perforant path. *J. Physiol.*, 1973, *232*, 331-356.

Broomhead, D.S. & Lowe, D. Multivariable functional interpolation and adaptive networks. *Complex Systems*, 1988, *2*, 321-355.

Brown, T.H., Chapman, P.F., Kairiss, E.W., & Keenan, C.L. Long-term synaptic potentiation. *Science*, 1988, *242*, 724-728.

Durbin, R. & Rumelhart, D.E. Product units: a computationally powerful and biologically plausible extension to backpropagation networks. *Complex Systems*, 1989, *1*, 133.

Eccles, J.C. The cerebral neocortex: a theory of its operation. In *Cerebral Cortex, vol. 2*, A. Peters & E.G. Jones, (Eds.), Plenum: New York, 1985.

Feldman, J.A. & Ballard, D.H. Connectionist models and their properties. *Cognitive Science*, 1982, *6*, 205-254.

Giles, C.L. & Maxwell, T. Learning, invariance, and generalization in high-order neural networks. *Applied Optics*, 1987, *26(23)*, 4972-4978.

Hebb, D.O. *The organization of behavior*. New York: Wiley, 1949.

Jones, E.G. Anatomy of cerebral cortex: columnar input-ouput relations. In *The organization of cerebral cortex*, F.O. Schmitt, F.G. Worden, G. Adelman, & S.G. Dennis, (Eds.), MIT Press: Cambridge, MA, 1981.

Kelso, S.R., Ganong, A.H., & Brown, T.H. Hebbian synapses in hippocampus. *PNAS* USA, 1986, *83*, 5326-5330.

Koch, C., Poggio, T., & Torre, V. Nonlinear interactions in a dendritic tree: localization, timing, and role in information processing. *PNAS*, 1983, *80*, 2799-2802.

Lapedes, A. & Farber, R. How neural nets work. In *Neural Information Processing Systems*, D.Z. Anderson, (Ed.), American Institute of Physics: New York, 1988.

Lund, J.S. Spiny stellate neurons. In *Cerebral Cortex, vol. 1*, A. Peters & E.G. Jones, (Eds.), Plenum: New York, 1985.

Marr, D. A theory for cerebral neocortex. *Proc. Roy. Soc. Lond. B*, 1970, *176*, 161-234.

Marr, D. A theory of cerebellar cortex. *J. Physiol.*, 1969, *202*, 437-470.

Mel, B.W. MURPHY: A robot that learns by doing. In *Neural Information Processing Systems*, D.Z. Anderson, (Ed.), American Institute of Physics: New York, 1988.

Mel, B.W. MURPHY: A neurally inspired connectionist approach to learning and performance in vision-based robot motion planning. Ph.D. thesis, University of Illinois, 1989.

Miller W.T., Hewes, R.P., Glanz, F.H., & Kraft, L.G. Real time dynamic control of an industrial manipulator using a neural network based learning controller. Technical Report, Dept. of Electrical and Computer Engineering, University of New Hampshire, 1988.

Moody, J. & Darken, C. Learning with localized receptive fields. In *Proc. 1988 Connectionist Models Summer School*, Morgan-Kaufmann, 1988.

Peters, A. Plenary address, 1989 Soc. Neurosc. Meeting, Phoenix, AZ.

Poggio, T. & Girosi, F. Learning, networks and approximation theory. *Science*, In press.

Rumelhart, D.E., Hinton, G.E., & McClelland, J.L. A general framework for parallel distributed processing. In *Parallel distributed processing: explorations in the microstructure of cognition, vol. 1*, D.E. Rumelhart, J.L. McClelland, (Eds.), Cambridge, MA: Bradford, 1986.

Shepherd, G.M., Brayton, R.K., Miller, J.P., Segev, I., Rinzel, J., & Rall, W. Signal enhancement in distal cortical dendrites by means of interactions between active dendritic spines. *PNAS*, 1985, *82*, 2192-2195.

Szentagothai, J. The neuron network of the cerebral cortex: a functional interpretation. (1977) *Proc. R. Soc. Lond. B.*, 201:219-248.